# Online $\ell_1$-Dictionary Learning with Application to Novel Document Detection

**Shiva Prasad Kasiviswanathan**[*]
General Electric Global Research
kasivisw@gmail.com

**Huahua Wang**[†]
University of Minnesota
huwang@cs.umn.edu

**Arindam Banerjee**[†]
University of Minnesota
banerjee@cs.umn.edu

**Prem Melville**
IBM T.J. Watson Research Center
pmelvil@us.ibm.com

## Abstract

Given their pervasive use, social media, such as Twitter, have become a leading source of breaking news. A key task in the automated identification of such news is the detection of novel documents from a voluminous stream of text documents in a scalable manner. Motivated by this challenge, we introduce the problem of online $\ell_1$-dictionary learning where unlike traditional dictionary learning, which uses squared loss, the $\ell_1$-penalty is used for measuring the reconstruction error. We present an efficient online algorithm for this problem based on alternating directions method of multipliers, and establish a sublinear regret bound for this algorithm. Empirical results on news-stream and Twitter data, shows that this online $\ell_1$-dictionary learning algorithm for novel document detection gives more than an order of magnitude speedup over the previously known batch algorithm, without any significant loss in quality of results.

## 1 Introduction

The high volume and velocity of social media, such as blogs and Twitter, have propelled them to the forefront as sources of breaking news. On Twitter, it is possible to find the latest updates on diverse topics, from natural disasters to celebrity deaths; and identifying such emerging topics has many practical applications, such as in marketing, disease control, and national security [14]. The key challenge in automatic detection of breaking news, is being able to detect novel documents in a stream of text; where a document is considered novel if it is "unlike" documents seen in the past. Recently, this has been made possible by *dictionary learning*, which has emerged as a powerful data representation framework. In dictionary learning each data point $\mathbf{y}$ is represented as a sparse linear combination $A\mathbf{x}$ of dictionary atoms, where $A$ is the dictionary and $\mathbf{x}$ is a sparse vector [1, 12]. A dictionary learning approach can be easily converted into a novel document detection method: let $A$ be a dictionary representing all documents till time $t-1$, for a new data document $\mathbf{y}$ arriving at time $t$, if one does not find a sparse combination $\mathbf{x}$ of the dictionary atoms, and the best reconstruction $A\mathbf{x}$ yields a large loss, then $\mathbf{y}$ clearly is not well represented by the dictionary $A$, and is hence novel compared to documents in the past. At the end of timestep $t$, the dictionary is updated to represent all the documents till time $t$.

Kasiviswanathan *et al.* [10] presented such a (batch) dictionary learning approach for detecting novel documents/topics. They used an $\ell_1$-penalty on the reconstruction error (instead of squared loss com-

---

[*]Part of this wok was done while the author was a postdoc at the IBM T.J. Watson Research Center.

[†]H. Wang and A. Banerjee was supported in part by NSF CAREER grant IIS-0953274, NSF grants IIS-0916750, 1029711, IIS-0812183, and NASA grant NNX12AQ39A.

monly used in the dictionary learning literature) as the $\ell_1$-penalty has been found to be more effective for text analysis (see Section 3). They also showed this approach outperforms other techniques, such as a nearest-neighbor approach popular in the related area of *First Story Detection* [16]. We build upon this work, by proposing an efficient algorithm for online dictionary learning with $\ell_1$-penalty. Our online dictionary learning algorithm is based on the *online alternating directions* method which was recently proposed by Wang and Banerjee [19] to solve online composite optimization problems with additional linear equality constraints. Traditional online convex optimization methods such as [25, 8, 5, 6, 22] require explicit computation of the subgradient making them computationally expensive to be applied in our high volume text setting, whereas in our algorithm the subgradients are computed implicitly. The algorithm has simple closed form updates for all steps yielding a fast and scalable algorithm for updating the dictionary. Under suitable assumptions (to cope with the non-convexity of the dictionary learning problem), we establish an $O(\sqrt{T})$ regret bound for the objective, matching the regret bounds of existing methods [25, 5, 6, 22]. Using this online algorithm for $\ell_1$-dictionary learning, we obtain an online algorithm for novel document detection, which we empirically validate on traditional news-streams as well as streaming data from Twitter. Experimental results show a substantial speedup over the batch $\ell_1$-dictionary learning based approach of Kasiviswanathan *et al.* [10], without a loss of performance in detecting novel documents.

**Related Work.** Online convex optimization is an area of active research and for a detailed survey on the literature we refer the reader to [18]. Online dictionary learning was recently introduced by Mairal *et al.* [12] who showed that it provides a scalable approach for handling large dynamic datasets. They considered an $\ell_2$-penalty and showed that their online algorithm converges to the minimum objective value in the stochastic case (i.e., with distributional assumptions on the data). However, the ideas proposed in [12] do not translate to the $\ell_1$-penalty. The problem of novel document/topics detection was also addressed by a recent work of Saha *et al.* [17], where they proposed a non-negative matrix factorization based approach for capturing evolving and novel topics. However, their algorithm operates over a sliding time window (does not have online regret guarantees) and works only for $\ell_2$-penalty.

## 2 Preliminaries

**Notation.** Vectors are always column vectors and are denoted by boldface letters. For a matrix $Z$ its norm, $\|Z\|_1 = \sum_{i,j} |z_{ij}|$ and $\|Z\|_F^2 = \sum_{ij} z_{ij}^2$. For arbitrary real matrices the standard inner product is defined as $\langle Y, Z \rangle = \mathrm{Tr}(Y^\top Z)$. We use $\Psi_{\max}(Z)$ to denote the largest eigenvalue of $Z^\top Z$. For a scalar $r \in \mathbb{R}$, let $\mathrm{sign}(r) = 1$ if $r > 0$, $-1$ if $r < 0$, and $0$ if $r = 0$. Define $\mathrm{soft}(r, T) = \mathrm{sign}(r) \cdot \max\{|r| - T, 0\}$. The operators sign and soft are extended to a matrix by applying it to every entry in the matrix. $\mathbf{0}_{m \times n}$ denotes a matrix of all zeros of size $m \times n$ and the subscript is omitted when the dimension of the represented matrix is clear from the context.

**Dictionary Learning Background.** *Dictionary learning* is the problem of estimating a collection of basis vectors over which a given data collection can be accurately reconstructed, often with sparse encodings. It falls into a general category of techniques known as *matrix factorization*. Classic dictionary learning techniques for sparse representation (see [1, 15, 12] and references therein) consider a finite training set of signals $P = [\mathbf{p}_1, \ldots, \mathbf{p}_n] \in \mathbb{R}^{m \times n}$ and optimize the empirical cost function which is defined as $f(A) = \sum_{i=1}^n l(\mathbf{p}_i, A)$, where $l(\cdot, \cdot)$ is a loss function such that $l(\mathbf{p}_i, A)$ should be small if $A$ is "good" at representing the signal $\mathbf{p}_i$ in a sparse fashion. Here, $A \in \mathbb{R}^{m \times k}$ is referred to as the *dictionary*. In this paper, we use a $\ell_1$-loss function with an $\ell_1$-regularization term, and our

$$l(\mathbf{p}_i, A) = \min_{\mathbf{x}} \ \|\mathbf{p}_i - A\mathbf{x}\|_1 + \lambda \|\mathbf{x}\|_1, \ \text{where } \lambda \text{ is the regularization parameter.}$$

We define the problem of dictionary learning as that of minimizing the empirical cost $f(A)$. In other words, the dictionary learning is the following optimization problem

$$\min_A f(A) = f(A, X) \overset{\text{def}}{=} \min_{A,X} \sum_{i=1}^n l(\mathbf{p}_i, A) = \min_{A,X} \ \|P - AX\|_1 + \lambda \|X\|_1.$$

For maintaining interpretability of the results, we would additionally require that the $A$ and $X$ matrices be non-negative. To prevent $A$ from being arbitrarily large (which would lead to arbitrarily small values of $X$), we add a scaling constant on $A$ as follows. Let $\mathcal{A}$ be the convex set of matrices defined as

$$\mathcal{A} = \{A \in \mathbb{R}^{m \times k} \ : \ A \geq \mathbf{0}_{m \times k} \ \forall j = 1, \ldots, k \,, \|A_j\|_1 \leq 1\}, \text{ where } A_j \text{ is the } j\text{th column in } A.$$

We use $\Pi_{\mathcal{A}}$ to denote the Euclidean projection onto the nearest point in the convex set $\mathcal{A}$. The resulting optimization problem can be written as

$$\min_{A \in \mathcal{A}, X \geq \mathbf{0}} \quad \|P - AX\|_1 + \lambda \|X\|_1 \tag{1}$$

The optimization problem (1) is in general non-convex. But if one of the variables, either $A$ or $X$ is known, the objective function with respect to the other variable becomes a convex function (in fact, can be transformed into a linear program).

## 3 Novel Document Detection Using Dictionary Learning

In this section, we describe the problem of novel document detection and explain how dictionary learning could be used to tackle this problem. Our problem setup is similar to [10].

**Novel Document Detection Task.** We assume documents arrive in streams. Let $\{P_t : P_t \in \mathbb{R}^{m_t \times n_t}, t = 1, 2, 3, \dots\}$ denote a sequence of streaming matrices where each column of $P_t$ represents a document arriving at time $t$. Here, $P_t$ represents the term-document matrix observed at time $t$. Each document is represented is some conventional vector space model such as TF-IDF [13]. The $t$ could be at any granularity, e.g., it could be the day that the document arrives. We use $n_t$ to represent the number of documents arriving at time $t$. We normalize $P_t$ such that each column (document) in $P_t$ has a unit $\ell_1$-norm. For simplicity in exposition, we will assume that $m_t = m$ for all $t$.[1] We use the notation $P_{[t]}$ to denote the term-document matrix obtained by vertically concatenating the matrices $P_1, \dots, P_t$, i.e., $P_{[t]} = [P_1|P_2|\dots|P_t]$. Let $N_t$ be the number of documents arriving at time $\leq t$, then $P_{[t]} \in \mathbb{R}^{m \times N_t}$. Under this setup, the *goal* of novel document detection is to identify documents in $P_t$ that are "dissimilar" to the documents in $P_{[t-1]}$.

**Sparse Coding to Detect Novel Documents.** Let $A_t \in \mathbb{R}^{m \times k}$ represent the dictionary matrix after time $t-1$; where dictionary $A_t$ is a good basis to represent of all the documents in $P_{[t-1]}$. The exact construction of the dictionary is described later. Now, consider a document $\mathbf{y} \in \mathbb{R}^m$ appearing at time $t$. We say that it admits a *sparse* representation over $A_t$, if $\mathbf{y}$ could be "well" approximated as a linear combination of few columns from $A_t$. Modeling a vector with such a sparse decomposition is known as *sparse coding*. In most practical situations it may not be possible to represent $\mathbf{y}$ as $A_t\mathbf{x}$, e.g., if $\mathbf{y}$ has new words which are absent in $A_t$. In such cases, one could represent $\mathbf{y} = A_t\mathbf{x} + \mathbf{e}$ where $\mathbf{e}$ is an unknown noise vector. We consider the following sparse coding formulation

$$l(\mathbf{y}, A_t) = \min_{\mathbf{x} \geq \mathbf{0}} \ \|\mathbf{y} - A_t\mathbf{x}\|_1 + \lambda\|\mathbf{x}\|_1. \tag{2}$$

The formulation (2) naturally takes into account both the reconstruction error (with the $\|\mathbf{y} - A_t\mathbf{x}\|_1$ term) and the complexity of the sparse decomposition (with the $\|\mathbf{x}\|_1$ term). It is quite easy to transform (2) into a linear program. Hence, it can be solved using a variety of methods. In our experiments, we use the alternating directions method of multipliers (ADMM) [2] to solve (2). ADMM has recently gathered significant attention in the machine learning community due to its wide applicability to a range of learning problems with complex objective functions [2].

We can use sparse coding to detect novel documents as follows. For each document $\mathbf{y}$ arriving at time $t$, we do the following. First, we solve (2) to check whether $\mathbf{y}$ could be well approximated as a sparse linear combination of the atoms of $A_t$. If the objective value $l(\mathbf{y}, A_t)$ is "big" then we mark the document as *novel*, otherwise we mark the document as *non-novel*. Since, we have normalized all documents in $P_t$ to unit $\ell_1$-length, the objective values are in the same scale.

**Choice of the Error Function.** A very common choice of reconstruction error is the $\ell_2$-penalty. In fact, in the presence of isotopic Gaussian noise the $\ell_2$-penalty on $\mathbf{e} = \mathbf{y} - A_t\mathbf{x}$ gives the maximum likelihood estimate of $\mathbf{x}$ [21, 23]. However, for text documents, the noise vector $\mathbf{e}$ rarely satisfies the Gaussian assumption, as some of its coefficients contain large, impulsive values. For example, in fields such as politics and sports, a certain term may become suddenly dominant in a discussion [10]. In such cases imposing an $\ell_1$-penalty on the error is a better choice than imposing an $\ell_2$-penalty (e.g., recent research [21, 24, 20] have successfully shown the superiority of $\ell_1$ over $\ell_2$ penalty for a

different but related application domain of face recognition). We empirically validate the superiority of using the $\ell_1$-penalty for novel document detection in Section 5.

**Size of the Dictionary.** Ideally, in our application setting, changing the size of the dictionary ($k$) dynamically with $t$ would lead to a more efficient and effective sparse coding. However, in our theoretical analysis, we make the simplifying assumption that $k$ is a constant independent of $t$. In our experiments, we allow for small increases in the size of the dictionary over time when required.

**Batch Algorithm for Novel Document Detection.** We now describe a simple batch algorithm (slightly modified from [10]) for detecting novel documents. The Algorithm BATCH alternates between a novel document detection and a batch dictionary learning step.

---

**Algorithm 1** : BATCH

---

**Input:** $P_{[t-1]} \in \mathbb{R}^{m \times N_{t-1}}$, $P_t = [\mathbf{p}_1, \ldots, \mathbf{p}_{n_t}] \in \mathbb{R}^{m \times n_t}$, $A_t \in \mathbb{R}^{m \times k}$, $\lambda \geq 0, \zeta \geq 0$
**Novel Document Detection Step:**
**for** $j = 1$ **to** $n_t$ **do**
    Solve: $\mathbf{x}_j = \mathrm{argmin}_{\mathbf{x} \geq 0} \|\mathbf{p}_j - A_t \mathbf{x}\|_1 + \lambda \|\mathbf{x}\|_1$
    **if** $\|\mathbf{p}_j - A_t \mathbf{x}_j\|_1 + \lambda \|\mathbf{x}_j\|_1 > \zeta$
        Mark $\mathbf{p}_j$ as novel
**Batch Dictionary Learning Step:**
Set $P_{[t]} \leftarrow [P_{[t-1]} \,|\, \mathbf{p}_1, \ldots, \mathbf{p}_{n_t}]$
Solve: $[A_{t+1}, X_{[t]}] = \mathrm{argmin}_{A \in \mathcal{A}, X \geq \mathbf{0}} \|P_{[t]} - AX\|_1 + \lambda \|X\|_1$

---

**Batch Dictionary Learning.** We now describe the batch dictionary learning step. At time $t$, the dictionary learning step is[2]

$$[A_{t+1}, X_{[t]}] = \mathrm{argmin}_{A \in \mathcal{A}, X \geq \mathbf{0}} \|P_{[t]} - AX\|_1 + \lambda \|X\|_1. \tag{3}$$

Even though conceptually simple, Algorithm BATCH is computationally inefficient. The bottleneck comes in the dictionary learning step. As $t$ increases, so does the size of $P_{[t]}$, so solving (3) becomes prohibitive even with efficient optimization techniques. To achieve computational efficiency, in [10], the authors solved an approximation of (3) where in the dictionary learning step they only update the $A$'s and not the $X$'s.[3] This leads to faster running times, but because of the approximation, the quality of the dictionary degrades over time and the performance of the algorithm decreases. In this paper, we propose an online learning algorithm for (3) and show that this online algorithm is both computationally efficient and generates good quality dictionaries under reasonable assumptions.

## 4 Online $\ell_1$-Dictionary Learning

In this section, we introduce the online $\ell_1$-dictionary learning problem and propose an efficient algorithm for it. The standard goal of online learning is to design algorithms whose regret is sublinear in time $T$, since this implies that "on the average" the algorithm performs as well as the best fixed strategy in hindsight [18]. Now consider the $\ell_1$-dictionary learning problem defined in (3). Since this problem is non-convex, it may not be possible to design *efficient* (i.e., polynomial running time) algorithms that solves it without making any assumptions on either the dictionary ($A$) or the sparse code ($X$). This also means that it may not be possible to design efficient online algorithm with sublinear regret without making any assumptions on either $A$ or $X$ because an efficient online algorithm with sublinear regret would imply an efficient algorithm for solving (1) in the offline case. Therefore, we focus on obtaining regret bounds for the dictionary update, assuming that the at each timestep the sparse codes given to the batch and online algorithms are "close". This motivates the following problem.

**Definition 4.1** (Online $\ell_1$-Dictionary Learning Problem). *At time t, the online algorithm picks* $\hat{A}_{t+1} \in \mathcal{A}$. *Then, the nature (adversary) reveals* $(P_{t+1}, \hat{X}_{t+1})$ *with* $P_{t+1} \in \mathbb{R}^{m \times n}$ *and* $\hat{X}_{t+1} \in$

$\mathbb{R}^{k \times n}$. *The problem is to pick the $\hat{A}_{t+1}$ sequence such that the following regret function is minimized*[4]

$$R(T) = \sum_{t=1}^{T} \|P_t - \hat{A}_t \hat{X}_t\|_1 - \min_{A \in \mathcal{A}} \sum_{t=1}^{T} \|P_t - AX_t\|_1 \ ,$$

*where $\hat{X}_t = X_t + E_t$ and $E_t$ is an error matrix dependent on $t$.*

The regret defined above admits the discrepancy between the sparse coding matrices supplied to the batch and online algorithms through the error matrix. The reason for this generality is because in our application setting, the sparse coding matrices used for updating the dictionaries of the batch and online algorithms could be different. We will later establish the conditions on $E_t$'s under which we can achieve sublinear regret. All missing proofs and details appear in the full version of the paper [11].

## 4.1 Online $\ell_1$-Dictionary Algorithm

In this section, we design an algorithm for the online $\ell_1$-dictionary learning problem, which we call Online Inexact ADMM (OIADMM)[5] and bound its regret. Firstly note that because of the non-smooth $\ell_1$-norms involved it is computationally expensive to apply standard online learning algorithms like online gradient descent [25, 8], COMID [6], FOBOS [5], and RDA [22], as they require computing a costly subgradient at every iteration. The subgradient of $\|P - AX\|_1$ at $A = \hat{A}$ is $(X \cdot \text{sign}(X^{\top} \hat{A}^{\top} - P^{\top}))^{\top}$.

Our algorithm for online $\ell_1$-dictionary learning is based on the online alternating direction method which was recently proposed by Wang *et al.* [19]. Our algorithm first performs a simple variable substitution by introducing an equality constraint. The update for each of the resulting variable has a closed-form solution without the need of estimating the subgradients explicitly.

---

**Algorithm 2** : OIADMM

---

**Input:** $P_t \in \mathbb{R}^{m \times n}$, $\hat{A}_t \in \mathbb{R}^{m \times k}$, $\Delta_t \in \mathbb{R}^{m \times n}$, $\hat{X}_t \in \mathbb{R}^{k \times n}$, $\beta_t \geq 0$, $\tau_t \geq 0$
$\widetilde{\Gamma}_t \longleftarrow P_t - \hat{A}_t \hat{X}_t$
$\Gamma_{t+1} = \text{argmin}_{\Gamma} \ \|\Gamma\|_1 + \langle \Delta_t, \widetilde{\Gamma}_t - \Gamma \rangle + (\beta_t/2)\|\widetilde{\Gamma}_t - \Gamma\|_F^2$
$\qquad\qquad\qquad\qquad\qquad\qquad\qquad (\Rightarrow \Gamma_{t+1} = \text{soft}(\widetilde{\Gamma}_t + \Delta_t/\beta_t, 1/\beta_t))$
$G_{t+1} \longleftarrow -(\Delta_t/\beta_t + \widetilde{\Gamma}_t - \Gamma_{t+1})\hat{X}_t^{\top}$
$\hat{A}_{t+1} = \text{argmin}_{A \in \mathcal{A}} \ \beta_t(\langle G_{t+1}, A - \hat{A}_t \rangle + (1/2\tau_t)\|A - \hat{A}_t\|_F^2)$
$\qquad\qquad\qquad\qquad\qquad\qquad\qquad (\Rightarrow \hat{A}_{t+1} = \Pi_{\mathcal{A}}(\max\{0, \hat{A}_t - \tau_t G_{t+1}\}))$
$\Delta_{t+1} = \Delta_t + \beta_t(P_t - \hat{A}_{t+1}\hat{X}_t - \Gamma_{t+1})$
Return $\hat{A}_{t+1}$ and $\Delta_{t+1}$

---

The Algorithm OIADMM is simple. Consider the following minimization problem at time $t$

$$\min_{A \in \mathcal{A}} \ \|P_t - A\hat{X}_t\|_1.$$

We can rewrite this above minimization problem as:

$$\min_{A \in \mathcal{A}, \Gamma} \ \|\Gamma\|_1 \quad \text{such that} \quad P_t - A\hat{X}_t = \Gamma. \qquad (4)$$

The augmented Lagrangian of (4) is:

$$\mathcal{L}(A, \Gamma, \Delta) \quad = \quad \min_{A \in \mathcal{A}, \Gamma} \quad \|\Gamma\|_1 \ + \ \langle \Delta, P_t \ - \ A\hat{X}_t \ - \ \Gamma \rangle \ + \ \frac{\beta_t}{2} \left\| P_t - A\hat{X}_t - \Gamma \right\|_F^2, \quad (5)$$

where $\Delta \in \mathbb{R}^{m \times n}$ is a multiplier and $\beta_t > 0$ is a penalty parameter.

OIADMM is summarized in Algorithm 2. The algorithm generates a sequence of iterates $\{\Gamma_t, A_t, \Delta_t\}_{t=1}^{\infty}$. At each time $t$, instead of solving (4) completely, it only runs one step ADMM update of the variables $(\Gamma_t, A_t, \Delta_t)$. The complete analysis of Algorithm 2 is presented in the full version of the paper [11]. Here, we just summarize the main result in the following theorem.

**Theorem 4.2.** *Let $\{\Gamma_t, \hat{A}_t, \Delta_t\}$ be the sequences generated by the OIADMM procedure and $R(T)$ be the regret as defined above. Assume the following conditions hold: (i) $\forall t$, the Frobenius norm of $\partial\|\Gamma_t\|_1$ is upper bounded by $\Phi$, (ii) $\hat{A}_1 = \mathbf{0}_{m \times k}$, $\|A^{\mathrm{opt}}\|_F \leq D$, (iii) $\Delta_1 = \mathbf{0}_{m \times n}$, and (iv) $\forall t$, $1/\tau_t \geq 2\Psi_{\max}(\hat{X}_t)$. Setting $\forall t$, $\beta_t = \frac{\Phi}{D}\sqrt{\tau_m T}$ where $\tau_m = \max_t\{1/\tau_t\}$, we have*

$$R(T) \leq \frac{\Phi D \sqrt{T}}{\sqrt{\tau_m}} + \sum_{t=1}^{T} \|A^{\mathrm{opt}} E_t\|_1.$$

In the above theorem one could replace $\tau_m$ by any upper bound on it (i.e., we don't need to know $\tau_m$ exactly).

**Condition on $E_t$'s for Sublinear Regret.** In a standard online learning setting, the $(P_t, \hat{X}_t)$ made available to the online learning algorithm will be the same as $(P_t, X_t)$ made available to the batch dictionary learning algorithm in hindsight, so that $\hat{X}_t = X_t \Rightarrow E_t = \mathbf{0}$, yielding a $O(\sqrt{T})$ regret. More generally, as long as $\sum_{t=1}^{T} \|E_t\|_p = o(T)$ for some suitable $p$-norm, we get a sublinear regret bound[6] For example, if $\{Z_t\}$ is a sequence of matrices such that for all $t$, $\|Z_t\|_p = O(1)$, then setting $E_t = t^{-\epsilon}Z_t, \epsilon > 0$ yields a sublinear regret. This gives a sufficient condition for sublinear regret, and it is an interesting open problem to extend the analysis to other cases.

**Running Time.** For the $i$th column in the dictionary matrix the projection onto $\mathcal{A}$ can be done in $O(s_i \log m)$ time where $s_i$ is the number of non-zero elements in the $i$th column using the projection onto $\ell_1$-ball algorithm of Duchi *et al.* [4]. The simplest implementation of OIADMM takes $O(mnk)$ time at each timestep because of the matrix multiplications involved.

## 5 Experimental Results

In this section, we present experiments to compare and contrast the performance of $\ell_1$-batch and $\ell_1$-online dictionary learning algorithms for the task of novel document detection. We also present results highlighting the superiority of using an $\ell_1$- over an $\ell_2$-penalty on the reconstruction error for this task (validating the discussion in Section 3).

**Implementation of** BATCH**.** In our implementation, we grow the dictionary size by $\eta$ in each timestep. Growing the dictionary size is essential for the batch algorithm because as $t$ increases the number of columns of $P_{[t]}$ also increases, and therefore, a larger dictionary is required to compactly represent all the documents in $P_{[t]}$. For solving (3), we use alternative minimization over the variables. The pseudo-code description is given in the full version of the paper [11]. The optimization problems arising in the sparse coding and dictionary learning steps are solved using ADMM's.

**Online Algorithm for Novel Document Detection.** Our online algorithm[7] uses the same novel document detection step as Algorithm BATCH, but dictionary learning is done using OIADMM. For a pseudo-code description, see full version of the paper [11]. Notice that the sparse coding matrices of the Algorithm BATCH, $X_1, \ldots, X_t$ could be different from $\hat{X}_1, \ldots, \hat{X}_t$. If these sequence of matrices are close to each other, then we have a sublinear regret on the objective function.[8]

**Evaluation of Novel Document Detection.** For performance evaluation, we assume that documents in the corpus have been manually identified with a set of topics. For simplicity, we assume that each document is tagged with the single, most dominant topic that it associates with, which we call the *true topic* of that document. We call a document $\mathbf{y}$ arriving at time $t$ *novel* if the true topic of $\mathbf{y}$ has not appeared before the time $t$. So at time $t$, given a set of documents, the task of novel

document detection is to classify each document as either novel (positive) or non-novel (negative). For evaluating this classification task, we use the standard Area Under the ROC Curve (AUC) [13].

**Performance Evaluation for $\ell_1$-Dictionary Learning.** We use a simple reconstruction error measure for comparing the dictionaries produced by our $\ell_1$-batch and $\ell_1$-online algorithms. We want the dictionary at time $t$ to be a good basis to represent all the documents in $P_{[t]} \in \mathbb{R}^{m \times N_t}$. This leads us to define the *sparse reconstruction error* (SRE) of a dictionary $A$ at time $t$ as

$$\text{SRE}(A) \stackrel{\text{def}}{=} \frac{1}{N_t} \left( \min_{X \geq \mathbf{0}} \|P_{[t]} - AX\|_1 + \lambda \|X\|_1 \right).$$

A dictionary with a smaller SRE is better on average at sparsely representing the documents in $P_{[t]}$.

**Novel Document Detection using $\ell_2$-dictionary learning.** To justify the choice of using an $\ell_1$-penalty (on the reconstruction error) for novel document detection, we performed experiments comparing $\ell_1$- vs. $\ell_2$-penalty for this task. In the $\ell_2$-setting, for the sparse coding step we used a fast implementation of the LARS algorithm with positivity constraints [7] and the dictionary learning was done by solving a non-negative matrix factorization problem with additional sparsity constraints (also known as the non-negative sparse coding problem [9]). A complete pseudo-code description is given in the full version of the paper [11].[9]

**Experimental Setup.** All reported results are based on a Matlab implementation running on a quad-core 2.33 GHz Intel processor with 32GB RAM. The regularization parameter $\lambda$ is set to 0.1 which yields reasonable sparsities in our experiments. OIADMM parameters $\tau_t$ are set $1/(2\Psi_{\max}(\hat{X}_t))$ (chosen according to Theorem 4.2) and $\beta_t$ is fixed to 5 (obtained through tuning). The ADMM parameters for the sparse coding and batch dictionary learning steps are set as suggested in [10] (refer to the full version [11]). In the batch algorithms, we grow the dictionary sizes by $\eta = 10$ in each timestep. The threshold value $\zeta$ is treated as a tunable parameter.

## 5.1 Experiments on News Streams

Our first dataset is drawn from the NIST Topic Detection and Tracking (TDT2) corpus which consists of news stories in the first half of 1998. In our evaluation, we used a set of 9000 documents represented over 19528 terms and distributed into the top 30 TDT2 human-labeled topics over a period of 27 weeks. We introduce the documents in groups. At timestep 0, we introduce the first 1000 documents and these documents are used for initializing the dictionary. We use an alternative minimization procedure over the variables of (1) to initialize the dictionary. In these experiments the size of the initial dictionary $k = 200$. In each subsequent timestep $t \in \{1, \ldots, 8\}$, we provide the batch and online algorithms the same set of 1000 documents. In Figure 1, we present novel document detection results for those timesteps where at least one novel document was introduced. Table 1 shows the corresponding AUC numbers. The results show that using an $\ell_1$-penalty on the reconstruction error is better for novel document detection than using an $\ell_2$-penalty.

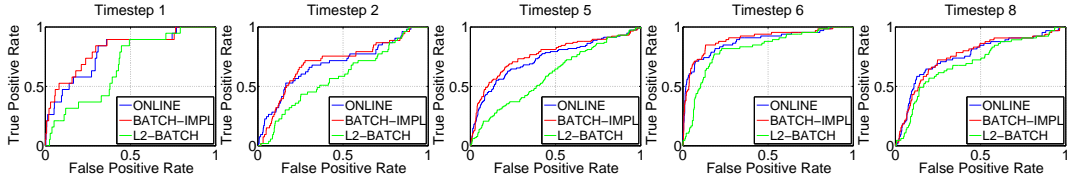

Figure 1: ROC curves for TDT2 for timesteps where novel documents were introduced.

**Comparison of the $\ell_1$-online and $\ell_1$-batch Algorithms.** The $\ell_1$-online and $\ell_1$-batch algorithms have almost identical performance in terms of detecting novel documents (see Table 1). However, the online algorithm is much more computationally efficient. In Figure 2(a), we compare the running times of these algorithms. As noted earlier, the running time of the batch algorithm goes up as $t$ increases (as it has to optimize over the entire past). However, the running time of the online algorithm is independent of the past and only depends on the number of documents introduced in each timestep (which in this case is always 1000). Therefore, the running time of the online

| Timestep | No. of Novel Docs. | No. of Nonnovel Docs. | AUC $\ell_1$-online | AUC $\ell_1$-batch | AUC $\ell_2$-batch |
|---|---|---|---|---|---|
| 1 | 19 | 981 | 0.791 | 0.815 | 0.674 |
| 2 | 53 | 947 | 0.694 | 0.704 | 0.586 |
| 5 | 116 | 884 | 0.732 | 0.764 | 0.601 |
| 6 | 66 | 934 | 0.881 | 0.898 | 0.816 |
| 8 | 65 | 935 | 0.757 | 0.760 | 0.701 |
| **Avg.** | | | **0.771** | **0.788** | **0.676** |

Table 1: AUC Numbers for ROC Plots in Figure 1.

algorithm is almost the same across different timesteps. As expected the run-time gap between the $\ell_1$-batch and $\ell_1$-online algorithms widen as $t$ increases – in the first timestep the online algorithm is $5.4$ times faster, and this rapidly increases to a factor of $11.5$ in just 7 timesteps.

In Figure 2(b), we compare the dictionaries produced by the $\ell_1$-batch and $\ell_1$-online algorithms under the SRE metric. In the first few timesteps, the SRE of the dictionaries produced by the online algorithm is slightly lower than that of the batch algorithm. However, this gets corrected after a few timesteps and as expected later on the batch algorithm produces better dictionaries.

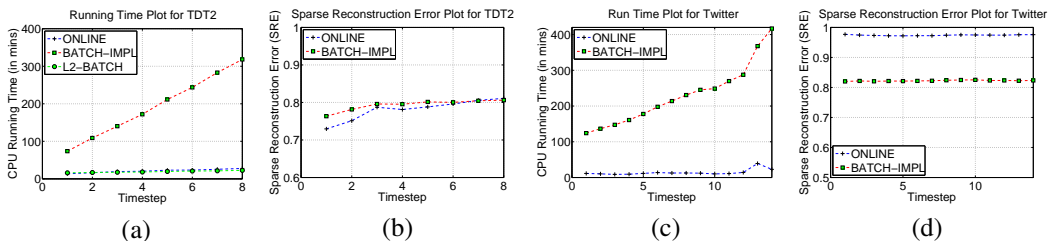

|     (a)     |     (b)     |     (c)     |     (d)     |

Figure 2: Running time and SRE plots for TDT2 and Twitter datasets.

## 5.2 Experiments on Twitter

Our second dataset is from an application of monitoring Twitter for Marketing and PR for smartphone and wireless providers. We used the Twitter Decahose to collect a $10\%$ sample of all tweets (posts) from Sept 15 to Oct 05, 2011. From this, we filtered the tweets relevant to "Smartphones" using a scheme presented in [3] which utilizes the Wikipedia ontology to do the filtering. Our dataset comprises of 127760 tweets over these 21 days and the vocabulary size is 6237 words. We used the tweets from Sept 15 to 21 (34292 in number) to initialize the dictionaries. Subsequently, at each timestep, we give as input to both the algorithms all the tweets from a given day (for a period of $14$ days between Sept 22 to Oct 05). Since this dataset is unlabeled, we do a quantitative evaluation of $\ell_1$-batch vs. $\ell_1$-online algorithms (in terms of SRE) and do a qualitative evaluation of the $\ell_1$-online algorithm for the novel document detection task. Here, the size of the initial dictionary $k = 100$.

Figure 2(c) shows the running times on the Twitter dataset. At first timestep the online algorithm is already $10.8$ times faster, and this speedup escalates to $18.2$ by the 14th timestep. Figure 2(d) shows the SRE of the dictionaries produced by these algorithms. In this case, the SRE of the dictionaries produced by the batch algorithm is consistently better than that of the online algorithm, but as the running time plots suggests this improvement comes at a very steep price.

| Date | Sample Novel Tweets Detected Using our Online Algorithm |
|---|---|
| 2011-09-26 | Android powered 56 percent of smartphones sold in the last three months. Sad thing is it can't lower the rating of ios! |
| 2011-09-29 | How Windows 8 is faster, lighter and more efficient: WP7 Droid Bionic Android 2.3.4 HP TouchPad white ipods_72 |
| 2011-10-03 | U.S. News: AT&T begins sending throttling warnings to top data hogs: AT&T did away with its unlimited da... #iPhone |
| 2011-10-04 | Can't wait for the iphone 4s #letstalkiphone |
| 2011-10-05 | Everybody put an iPhone up in the air one time #ripstevejobs |

Table 2: Sample novel documents detected by our online algorithm.

Table 2 below shows a representative set of novel tweets identified by our online algorithm. Using a completely automated process (refer to the full version [11]), we are able to detect breaking news and trending relevant to the smartphone market, such as AT&T throttling data bandwidth, launch of IPhone 4S, and the death of Steve Jobs.

## Footnotes

[1]As new documents come in and new terms are identified, we expand the vocabulary and zero-pad the previous matrices so that at the current time $t$, all previous and current documents have a representation over the same vocabulary space.

[2]In our algorithms, it is quite straightforward to replace the condition $A \in \mathcal{A}$ by some other condition $A \in \mathcal{C}$, where $\mathcal{C}$ is some closed non-empty convex set.

[3]In particular, define (recursively) $\widetilde{X}_{[t]} = [\widetilde{X}_{[t-1]} \,|\, \mathbf{x}_1, \ldots, \mathbf{x}_{n_t}]$ where $\mathbf{x}_j$'s are coming from the novel document detection step at time $t$. In [10], the dictionary learning step is $A_{t+1} = \mathrm{argmin}_{A \in \mathcal{A}} \|P_{[t]} - A\widetilde{X}_{[t]}\|_1$.

[4]For ease of presentation and analysis, we will assume that $m$ and $n$ don't vary with time. One could allow for changing $m$ and $n$ by carefully adjusting the size of the matrices by zero-padding.

[5]The reason for naming it OIADMM is because the algorithm is based on alternating directions method of multipliers (ADMM) procedure.

[6]This follows from Hölder's inequality which gives $\sum_{t=1}^{T}\|A^{\mathrm{opt}}E_t\|_1 \leq \|A^{\mathrm{opt}}\|_q (\sum_{t=1}^{T}\|E_t\|_p)$ for $1 \leq p, q \leq \infty$ and $1/p + 1/q = 1$, and by the assuming $\|A^{\mathrm{opt}}\|_q$ is bounded. Here, $\|\cdot\|_p$ denotes Schatten $p$-norm.

[7]In our experiments, the number of documents introduced in each timestep is almost of the same order, and hence there is no need to change the size of the dictionary across timesteps for the online algorithm.

[8]As noted earlier, we can not do a comparison without making any assumptions.

[9]We used the SPAMS package http://spams-devel.gforge.inria.fr/ in our implementation.

# References

[1] M. Aharon, M. Elad, and A. Bruckstein. The K-SVD: An Algorithm for Designing Overcomplete Dictionaries for Sparse Representation. *IEEE Transactions on Signal Processing*, 54(11), 2006.

[2] S. Boyd, N. Parikh, E. Chu, B. Peleato, and J. Eckstein. Distributed Optimization and Statistical Learning via the Alternating Direction Method of Multipliers. *Foundations and Trends in Machine Learning*, 2011.

[3] V. Chenthamarakshan, P. Melville, V. Sindhwani, and R. D. Lawrence. Concept Labeling: Building Text Classifiers with Minimal Supervision. In *IJCAI*, pages 1225–1230, 2011.

[4] J. Duchi, S. Shalev-Shwartz, Y. Singer, and T. Chandra. Efficient Projections onto the l1-ball for Learning in High Dimensions. In *ICML*, pages 272–279, 2008.

[5] J. Duchi and Y. Singer. Efficient Online and Batch Learning using Forward Backward Splitting. *JMLR*, 10:2873–2898, 2009.

[6] J. C. Duchi, S. Shalev-Shwartz, Y. Singer, and A. Tewari. Composite Objective Mirror Descent. In *COLT*, pages 14–26, 2010.

[7] J. Friedman, T. Hastie, H. Hfling, and R. Tibshirani. Pathwise Coordinate Optimization. *The Annals of Applied Statistics*, 1(2):302–332, 2007.

[8] E. Hazan, A. Agarwal, and S. Kale. Logarithmic Regret Algorithms for Online Convex Optimization. *Machine Learning*, 69(2-3):169–192, 2007.

[9] P. O. Hoyer. Non-Negative Sparse Coding. In *IEEE Workshop on Neural Networks for Signal Processing*, pages 557–565, 2002.

[10] S. P. Kasiviswanathan, P. Melville, A. Banerjee, and V. Sindhwani. Emerging Topic Detection using Dictionary Learning. In *CIKM*, pages 745–754, 2011.

[11] S. P. Kasiviswanathan, H. Wang, A. Banerjee, and P. Melville. Online $\ell_1$-Dictionary Learning with Application to Novel Document Detection. `http://www.cse.psu.edu/~kasivisw/fullonlinedict.pdf`.

[12] J. Mairal, F. Bach, J. Ponce, and G. Sapiro. Online Learning for Matrix Factorization and Sparse Coding. *JMLR*, 11:19–60, 2010.

[13] C. Manning, P. Raghavan, and H. Schütze. *Introduction to Information Retrieval*. Cambridge University Press, 2008.

[14] P. Melville, J. Leskovec, and F. Provost, editors. *Proceedings of the First Workshop on Social Media Analytics*. ACM, 2010.

[15] B. Olshausen and D. Field. Sparse Coding with an Overcomplete Basis Set: A Strategy Employed by V1? *Vision Research*, 37(23):3311–3325, 1997.

[16] S. Petrović, M. Osborne, and V. Lavrenko. Streaming First Story Detection with Application to Twitter. In *HLT '10*, pages 181–189. ACL, 2010.

[17] A. Saha and V. Sindhwani. Learning Evolving and Emerging Topics in Social Media: A Dynamic NMF Approach with Temporal Regularization. In *WSDM*, pages 693–702, 2012.

[18] S. Shalev-Shwartz. Online Learning and Online Convex Optimization. *Foundations and Trends in Machine Learning*, 4(2), 2012.

[19] H. Wang and A. Banerjee. Online Alternating Direction Method. In *ICML*, 2012.

[20] J. Wright and Y. Ma. Dense Error Correction Via L1-Minimization. *IEEE Transactions on Information Theory*, 56(7):3540–3560, 2010.

[21] J. Wright, A. Yang, A. Ganesh, S. Sastry, and Y. Ma. Robust Face Recognition via Sparse Representation. *IEEE Transactions on Pattern Analysis and Machine Intelliegence*, 31(2):210–227, Feb. 2009.

[22] L. Xiao. Dual Averaging Methods for Regularized Stochastic Learning and Online Optimization. *JMLR*, 11:2543–2596, 2010.

[23] A. Y. Yang, S. S. Sastry, A. Ganesh, and Y. Ma. Fast L1-minimization Algorithms and an Application in Robust Face Recognition: A Review. In *International Conference on Image Processing*, pages 1849–1852, 2010.

[24] J. Yang and Y. Zhang. Alternating Direction Algorithms for L1-Problems in Compressive Sensing. *SIAM Journal of Scientific Computing*, 33(1):250–278, 2011.

[25] M. Zinkevich. Online Convex Programming and Generalized Infinitesimal Gradient Ascent. In *ICML*, pages 928–936, 2003.

